# Learning long–term dependencies is not as difficult with NARX networks

**Tsungnan Lin**[*]
Department of Electrical Engineering
Princeton University
Princeton, NJ 08540

**Bill G. Horne**
NEC Research Institute
4 Independence Way
Princeton, NJ 08540

**Peter Tiňo**
Dept. of Computer Science and Engineering
Slovak Technical University
Ilkovicova 3, 812 19 Bratislava, Slovakia

**C. Lee Giles**[†]
NEC Research Institute
4 Independence Way
Princeton, NJ 08540

## Abstract

It has recently been shown that gradient descent learning algorithms for recurrent neural networks can perform poorly on tasks that involve long–term dependencies. In this paper we explore this problem for a class of architectures called NARX networks, which have powerful representational capabilities. Previous work reported that gradient descent learning is more effective in NARX networks than in recurrent networks with "hidden states". We show that although NARX networks do not circumvent the problem of long–term dependencies, they can greatly improve performance on such problems. We present some experimental results that show that NARX networks can often retain information for two to three times as long as conventional recurrent networks.

## 1 Introduction

Recurrent Neural Networks (RNNs) are capable of representing arbitrary nonlinear dynamical systems [19, 20]. However, learning simple behavior can be quite

---

[*]Also with NEC Research Institute.
[†]Also with UMIACS, University of Maryland, College Park, MD 20742

difficult using gradient descent. For example, even though these systems are Turing equivalent, it has been difficult to get them to successfully learn small finite state machines from example strings encoded as temporal sequences. Recently, it has been demonstrated that at least part of this difficulty can be attributed to *long-term dependencies*, i.e. when the desired output at time $T$ depends on inputs presented at times $t \ll T$. In [13] it was reported that RNNs were able to learn short term musical structure using gradient based methods, but had difficulty capturing global behavior. These ideas were recently formalized in [2], which showed that if a system is to robustly latch information, then the fraction of the gradient due to information $n$ time steps in the past approaches zero as $n$ becomes large.

Several approaches have been suggested to circumvent this problem. For example, gradient-based methods can be abandoned in favor of alternative optimization methods [2, 15]. However, the algorithms investigated so far either perform just as poorly on problems involving long-term dependencies, or, when they are better, require far more computational resources [2]. Another possibility is to modify conventional gradient descent by more heavily weighing the fraction of the gradient due to information far in the past, but there is no guarantee that such a modified algorithm would converge to a minima of the error surface being searched [2]. Another suggestion has been to alter the input data so that it represents a reduced description that makes global features more explicit and more readily detectable [7, 13, 16, 17]. However, this approach may fail if short term dependencies are equally as important. Finally, it has been suggested that a network architecture that operates on multiple time scales might be useful [5, 6].

In this paper, we also propose an architectural approach to deal with long-term dependencies [11]. We focus on a class of architectures based upon Nonlinear AutoRegressive models with eXogenous inputs (NARX models), and are therefore called *NARX networks* [3, 14]. This is a powerful class of models which has recently been shown to be computationally equivalent to Turing machines [18]. Furthermore, previous work has shown that gradient descent learning is more effective in NARX networks than in recurrent network architectures with "hidden states" when applied to problems including grammatical inference and nonlinear system identification [8]. Typically, these networks converge much faster and generalize better than other networks. The results in this paper give an explanation of this phenomenon.

## 2   Vanishing gradients and long-term dependencies

Bengio *et al.* [2] have analytically explained why learning problems with long-term dependencies is difficult. They argue that for many practical applications the goal of the network must be to *robustly latch information*, i.e. the network must be able to store information for a long period of time in the presence of noise. More specifically, they argue that latching of information is accomplished when the states of the network stay within the vicinity of a hyperbolic attractor, and robustness to noise is accomplished if the states of the network are contained in the *reduced attracting set* of that attractor, i.e. those set of points at which the eigenvalues of the Jacobian are contained within the unit circle.

In algorithms such as Backpropagation Through Time (BPTT), the gradient of the cost function function C is written assuming that the weights at different time

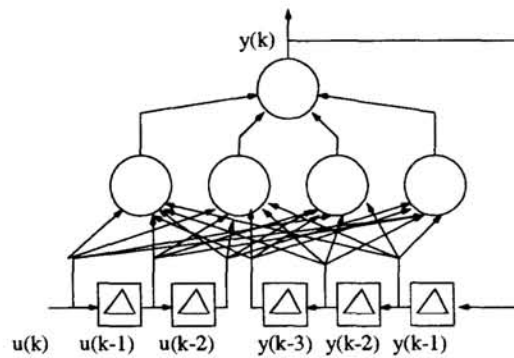

y(k)

u(k)   u(k-1)   u(k-2)   y(k-3)  y(k-2)  y(k-1)

Figure 1: NARX network.

indices are independent and computing the partial gradient with respect to these weights. The total gradient is then equal to the sum of these partial gradients.

It can be easily shown that the weight updates are proportional to

$$\nabla_{\mathbf{w}} C = \sum_p \left(\mathbf{y}_p(T) - \mathbf{d}_p\right) \nabla_{\mathbf{x}(T)} \mathbf{y}_p(T) \left[\sum_{\tau=1}^{T} J_{\mathbf{x}}(T, T - \tau) \nabla_{\mathbf{w}(\tau)} \mathbf{x}(\tau)\right],$$

where $\mathbf{y}_p(T)$ and $\mathbf{d}_p$ are the actual and desired (or target) output for the $p$th pattern[1], $\mathbf{x}(t)$ is the state vector of the network at time $t$ and $J_{\mathbf{x}}(T, T - \tau) = \nabla_{\mathbf{x}(\tau)} \mathbf{x}(T)$ denotes the Jacobian of the network expanded over $T - \tau$ time steps.

In [2], it was shown that if the network robustly latches information, then $J_{\mathbf{x}}(T, n)$ is an exponentially decreasing function of $n$, so that $\lim_{n \to \infty} J_{\mathbf{x}}(T, n) = 0$ . This implies that the portion of $\nabla_{\mathbf{w}} C$ due to information at times $\tau \ll T$ is insignificant compared to the portion at times near $T$. This vanishing gradient is the essential reason why gradient descent methods are not sufficiently powerful to discover a relationship between target outputs and inputs that occur at a much earlier time.

## 3   NARX networks

An important class of discrete–time nonlinear systems is the *Nonlinear AutoRegressive with eXogenous inputs* (NARX) model [3, 10, 12, 21]:

$$y(t) = f\left(u(t - D_u), \dots, u(t - 1), u(t), y(t - D_y), \dots, y(t - 1)\right),$$

where $u(t)$ and $y(t)$ represent input and output of the network at time $t$, $D_u$ and $D_y$ are the input and output order, and $f$ is a nonlinear function. When the function $f$ can be approximated by a Multilayer Perceptron, the resulting system is called a *NARX network* [3, 14].

In this paper we shall consider NARX networks with zero input order and a one dimensional output. However there is no reason why our results could not be extended to networks with higher input orders. Since the states of a discrete–time

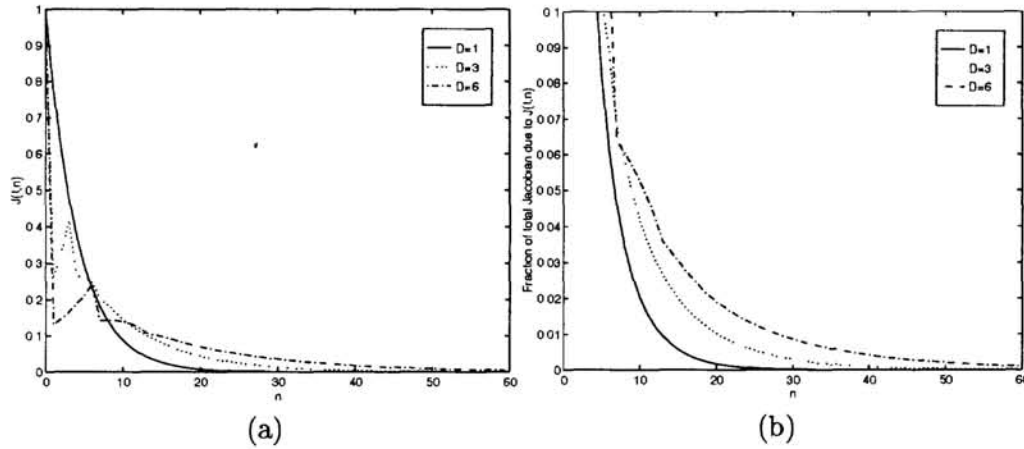

Figure 2: Results for the latching problem. (a) Plots of $J(t, n)$ as a function of $n$. (b) Plots of the ratio $\frac{J(t,n)}{\sum_{\tau=1}^{n} J(t,\tau)}$ as a function of $n$.

dynamical system can always be associated with the unit–delay elements in the realization of the system, we can then describe such a network in a state space form

$$x_i(t+1) = \begin{cases} \Psi\left(u(t), \mathbf{x}(t)\right) & i = 1 \\ x_{i-1}(t) & i = 2, \dots, D \end{cases} \tag{1}$$

with $y(t) = x_1(t+1)$ .

If the Jacobian of this system has all of its eigenvalues inside the unit circle at each time step, then the states of the network will be in the reduced attracting set of some hyperbolic attractor, and thus the system will be robustly latched at that time. As with any other RNN, this implies that $\lim_{n\to\infty} J_{\mathbf{x}}(t, n) = 0$. Thus, NARX networks will also suffer from vanishing gradients and the long–term dependencies problem. However, we find in the simulation results that follow that NARX networks are often much better at discovering long–term dependencies than conventional RNNs.

An intuitive reason why output delays can help long–term dependencies can be found by considering how gradients are calculated using the Backpropagation Through Time algorithm. BPTT involves two phases: unfolding the network in time and backpropagating the error through the unfolded network. When a NARX network is unfolded in time, the output delays will appear as jump–ahead connections in the unfolded network. Intuitively, these jump–ahead connections provide a shorter path for propagating gradient information, thus reducing the sensitivity of the network to long–term dependencies. However, this intuitive reasoning is only valid if the total gradient through these jump–ahead pathways is greater than the gradient through the layer–to–layer pathways.

It is possible to derive analytical results for some simple toy problems to show that NARX networks are indeed less sensitive to long–term dependencies. Here we give one such example, which is based upon the latching problem described in [2]. Consider the one node autonomous recurrent network described by, $x(t) = \tanh(wx(t-1))$ where $w = 1.25$, which has two stable fixed points at $\pm 0.710$ and one unstable fixed point at zero. The one node, autonomous NARX network $x(t) = \tanh\left(\sum_{\tau=1}^{D} w_\tau x(t-\tau)\right)$ has the same fixed points as long as $\sum_{i=1}^{D} w_i = w$.

Assume the state of the network has reached equilibrium at the positive stable fixed point and there are no external inputs. For simplicity, we only consider the Jacobian $J(t,n) = \frac{\partial x(t)}{\partial x(t-n)}$, which will be a component of the gradient $\nabla_{\mathbf{w}} C$. Figure 2a shows plots of $J(t,n)$ with respect to $n$ for $D = 1$, $D = 3$ and $D = 6$ with $w_i = w/D$. These plots show that the effect of output delays is to flatten out the curves and place more emphasis on the gradient due to terms farther in the past. Note that the gradient contribution due to short term dependencies is deemphasized. In Figure 2b we show plots of the ratio $\frac{J(t,n)}{\sum_{\tau=1}^{n} J(t,\tau)}$, which illustrates the percentage of the total gradient that can be attributed to information $n$ time steps in the past. These plots show that this percentage is larger for the network with output delays, and thus one would expect that these networks would be able to more effectively deal with long–term dependencies.

## 4 Experimental results

### 4.1 The latching problem

We explored a slight modification on the latching problem described in [2], which is a minimal task designed as a test that must necessarily be passed in order for a network to robustly latch information. In this task there are three inputs $u_1(t)$, $u_2(t)$, and a noise input $e(t)$, and a single output $y(t)$. Both $u_1(t)$ and $u_2(t)$ are zero for all times $t > 1$. At time $t = 1$, $u_1(1) = 1$ and $u_2(1) = 0$ for samples from class 1, and $u_1(1) = 0$ and $u_2(1) = 1$ for samples from class 2. The noise input $e(t)$ is drawn uniformly from $[-b, b]$ when $L < t \leq T$, otherwise $e(t) = 0$ when $t \leq L$. This network used to solve this problem is a NARX network consisting of a single neuron,

$$x(t) = \tanh\left(\sum_{\tau=1}^{D} w_\tau x(t-\tau) + \sum_{i=1}^{3} h_i^1 u_1(t-i+1) + \sum_{i=1}^{3} h_i^2 u_2(t-i+1) + e(t)\right)$$

where the parameters $h_i^j$ are adjustable and the recurrent weights $w_\tau$ are fixed [2].

We fixed the recurrent feedback weight to $w_\tau = 1.25/D$, which gives the autonomous network two stable fixed points at $\pm 0.710$, as described in Section 3. It can be shown [4] that the network is robust to perturbations in the range $[-0.155, 0.155]$. Thus, the uniform noise in $e(t)$ was restricted to this range.

For each simulation, we generated 30 strings from each class, each with a different $e(t)$. The initial values of $h_i^j$ for each simulation were also chosen from the same distribution that defines $e(t)$. For strings from class one, a target value of 0.8 was chosen, for class two $-0.8$ was chosen. The network was run using a simple BPTT algorithm with a learning rate of 0.1 for a maximum of 100 epochs. (We found that the network converged to some solution consistently within a few dozen epochs.) If the simulation exceeded 100 epochs and did not correctly classify all strings then the simulation was ruled a failure. We varied $T$ from 10 to 200 in increments of 2. For each value of $T$, we ran 50 simulations. Figure 3a shows a plot of the percentage of those runs that were successful for each case. It is clear from these plots that

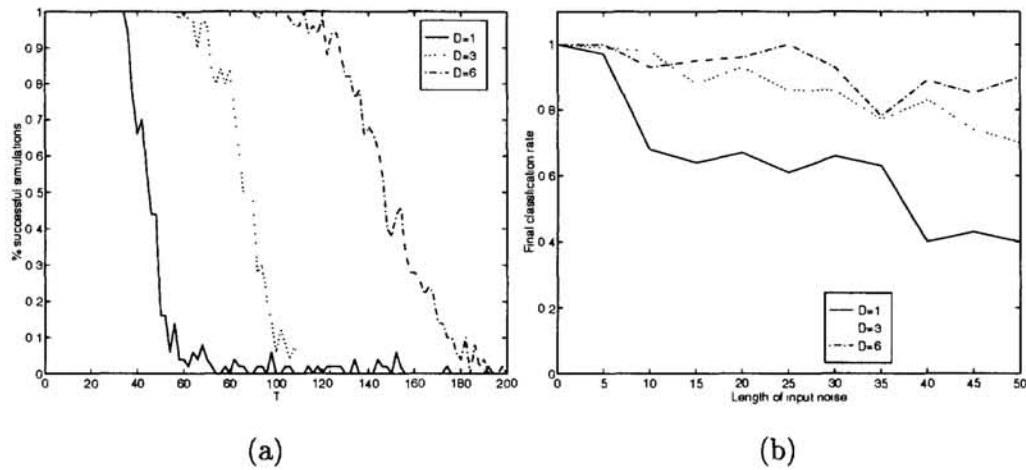

$$(a) \qquad\qquad\qquad\qquad (b)$$

Figure 3: (a) Plots of percentage of successful simulations as a function of $T$, the length of the input strings. (b) Plots of the final classification rate with respect to different length input strings.

the NARX networks become increasingly less sensitive to long–term dependencies as the output order is increased.

## 4.2   The parity problem

In the parity problem, the task is to classify sequences depending on whether or not the number of 1s in the input string is odd. We generated 20 strings of different lengths from 3 to 5 and added uniformly distributed noise in the range $[-0.2, 0.2]$ at the end of each string. The length of input noise varied from 0 to 50. We arbitrarily chose 0.7 and $-0.7$ to represent the symbol "1" and "0". The target is only given at the end of each string. Three different networks with different number of output delays were run on this problem in order to evaluate the capability of the network to learn long–term dependencies. In order to make the networks comparable, we chose networks in which the number of weights was roughly equal. For networks with one to three delays, 5, 4 and 3 hidden neurons were chosen respectively, giving 21, 21, and 19 trainable weights. Initial weight values were randomly generated between $-0.5$ and 0.5 for 10 trials.

Fig. 3b shows the average classification rate with respect to different length of input noise. When the length of the noise is less than 5, all three of the networks can learn all the sequences with the classification rate near to 100%. When the length increases to between 10 and 35, the classification rate of networks with one feedback delay drops quickly to about 60% while the rate of those networks with two or three feedback delays still remains about 80%.

## 5   Conclusion

In this paper we considered an architectural approach to dealing with the problem of learning long–term dependencies. We explored the ability of a class of architectures called NARX networks to solve such problems. This has been observed previously, in the sense that gradient descent learning appeared to be more effective in NARX

networks than in RNNs [8]. We presented an analytical example that showed that the gradients do not vanish as quickly in NARX networks as they do in networks without multiple delays when the network is operating at a fixed point. We also presented two experimental problems which show that NARX networks can out-perform networks with single delays on some simple problems involving long–term dependencies.

We speculate that similar results could be obtained for other networks. In particular we hypothesize that any network that uses tapped delay feedback [1, 9] would demonstrate improved performance on problems involving long–term dependencies.

## Acknowledgements

We would like to thank A. Back and Y. Bengio for many useful suggestions.

## Footnotes

[1]We deal only with problems in which the target output is presented at the *end* of the sequence.

[2] Although this description may appear different from the one in [2], it can be shown that they are actually identical experiments for $D = 1$.

## References

[1] A.D. Back and A.C. Tsoi. FIR and IIR synapses, a new neural network architecture for time series modeling. *Neural Computation*, 3(3):375–385, 1991.

[2] Y. Bengio, P. Simard, and P. Frasconi. Learning long-term dependencies with gradient is difficult. *IEEE Trans. on Neural Networks*, 5(2):157–166, 1994.

[3] S. Chen, S.A. Billings, and P.M. Grant. Non–linear system identification using neural networks. *International Journal of Control*, 51(6):1191–1214, 1990.

[4] P. Frasconi, M. Gori, M. Maggini, and G. Soda. Unified integration of explicit knowledge and learning by example in recurrent networks. *IEEE Trans. on Know. and Data Eng.*,7(2):340-346, 1995.

[5] M. Gori, M. Maggini, and G. Soda. Scheduling of modular architectures for inductive inference of regular grammars. In *ECAI'94 Work. on Comb. Sym. and Connectionist Proc.*, pages 78–87.

[6] S. El Hihi and Y. Bengio. Hierarchical recurrent neural networks for long-term dependencies. In *NIPS 8*, 1996. (In this Proceedings.)

[7] S. Hochreiter and J. Schmidhuber. Long short term memory. Technical Report FKI-207-95, Technische Universität München, 1995.

[8] B.G. Horne and C.L. Giles. An experimental comparison of recurrent neural networks. In *NIPS 7*, pages 697-704, 1995.

[9] R.R. Leighton and B.C. Conrath. The autoregressive backpropagation algorithm. In *Proceedings of the International Joint Conference on Neural Networks*, volume 2, pages 369–377, July 1991.

[10] I.J. Leontaritis and S.A. Billings. Input–output parametric models for non–linear systems: Part I: deterministic non–linear systems. *International Journal of Control*, 41(2):303–328, 1985.

[11] T.N. Lin, B.G. Horne, P.Tino and C.L. Giles. Learning long-term dependencies is not as difficult with NARX recurrent neural networks. Technical Report UMIACS-TR-95-78 and CS-TR-3500, Univ. Of Maryland, 1995.

[12] L. Ljung. *System identification: Theory for the user*. Prentice-Hall, 1987.

[13] M. C. Mozer. Induction of multiscale temporal structure. In J.E. Moody, S. J. Hanson, and R.P. Lippmann, editors, *NIPS 4*, pages 275–282, 1992.

[14] K.S. Narendra and K. Parthasarathy. Identification and control of dynamical systems using neural networks. *IEEE Trans. on Neural Networks*, 1:4–27, March 1990.

[15] G.V. Puskorius and L.A. Feldkamp. Recurrent network training with the decoupled extended Kalman filter. In *Proc. 1992 SPIE Conf. on the Sci. of ANN*, Orlando, Florida, April 1992.

[16] J. Schmidhuber. Learning complex, extended sequences using the principle of history compression. In *Neural Computation*, 4(2):234-242, 1992.

[17] J. Schmidhuber. Learning unambiguous reduced sequence descriptions. In *NIPS 4*, pages 291–298, 1992.

[18] H.T. Siegelmann, B.G. Horne, and C.L. Giles. Computational capabilities of NARX neural networks. In *IEEE Trans. on Systems, Man and Cybernetics*, 1996. Accepted.

[19] H.T. Siegelmann and E.D. Sontag. On the computational power of neural networks. *Journal of Computer and System Science*, 50(1):132–150, 1995.

[20] E.D. Sontag. Systems combining linearity and saturations and relations to neural networks. Technical Report SYCON–92–01, Rutgers Center for Systems and Control, 1992.

[21] H. Su, T. McAvoy, and P. Werbos. Long-term predictions of chemical processes using recurrent neural networks: A parallel training approach. *Ind. Eng. Chem. Res.*, 31:1338, 1992.
